# Finite-dimensional approximation of Gaussian processes

**Giancarlo Ferrari Trecate**
Dipartimento di Informatica e Sistemistica, Università di Pavia,
Via Ferrata 1, 27100 Pavia, Italy
ferrari@conpro.unipv.it

**Christopher K. I. Williams**
Department of Artificial Intelligence, University of Edinburgh,
5 Forrest Hill, Edinburgh EH1 2QL,
ckiw@dai.ed.ac.uk.

**Manfred Opper**
Neural Computing Research Group
Division of Electronic Engineering and Computer Science
Aston University, Birmingham, B4 7ET, UK
m.opper@aston.ac.uk

## Abstract

Gaussian process (GP) prediction suffers from $O(n^3)$ scaling with the data set size $n$. By using a finite-dimensional basis to approximate the GP predictor, the computational complexity can be reduced. We derive optimal finite-dimensional predictors under a number of assumptions, and show the superiority of these predictors over the Projected Bayes Regression method (which is asymptotically optimal). We also show how to calculate the minimal model size for a given $n$. The calculations are backed up by numerical experiments.

## 1 Introduction

Over the last decade there has been a growing interest in the Bayesian approach to regression problems, using both neural networks and Gaussian process (GP) prediction, that is regression performed in function spaces when using a Gaussian random process as a prior.

The computational complexity of the GP predictor scales as $O(n^3)$, where $n$ is the size

of the dataset[1]. This suggests using a finite-dimensional approximating function space, which we will assume has dimension $m < n$. The use of the finite-dimensional model is motivated by the need for regression algorithms computationally cheaper than the GP one. Moreover, GP regression may be used for the identification of dynamical systems (De Nicolao and Ferrari Trecate, 1998), the next step being a model-based controller design. In many cases it is easier to accomplish this second task if the model is low dimensional.

Use of a finite-dimensional model leads naturally to the question as to which basis is optimal. Zhu *et al.* (1997) show that, in the asymptotic régime, one should use the first $m$ eigenfunctions of the covariance function describing the Gaussian process. We call this method Projected Bayes Regression (PBR).

The main results of the paper are:

1. Although PBR is asymptotically optimal, for finite data we derive a predictor $h^o(x)$ with computational complexity $O(n^2 m)$ which outperforms PBR, and obtain an upper bound on the generalization error of $h^o(x)$.

2. In practice we need to know how large to make $m$. We show that this depends on $n$ and provide a means of calculating the minimal $m$. We also provide empirical results to back up the theoretical calculation.

## 2    Problem statement

Consider the problem of estimating an unknown function $f(x) : \mathbb{R}^d \to \mathbb{R}$, from the noisy observations

$$t_i = f(x_i) + \epsilon_i, \quad i = 1, \ldots, n$$

where $\epsilon_i$ are i.i.d. zero-mean Gaussian random variables with variance $\sigma^2$ and the samples $x_i$ are drawn independently at random from a distribution $p(x)$. The prior probability measure over the function $f(\cdot)$ is assumed to be Gaussian with zero mean and autocovariance function $C(\xi_1, \xi_2)$. Moreover we suppose that $f(\cdot)$, $x_i$, $\epsilon_i$, are mutually independent. Given the data set $\mathcal{D}_n = \{\bar{x}, \bar{t}\}$, where $\bar{x} = [x_1, \ldots, x_n]$ and $\bar{t} = [t_1, \ldots, t_n]'$, it is well known that the posterior probability $P(f|\mathcal{D}_n)$ is Gaussian and the GP prediction can be computed via explicit formula (e.g. Whittle, 1963)

$$\hat{f}(x) = \mathrm{E}[f|\mathcal{D}_n](x) = \begin{bmatrix} C(x, x_1) & \cdots & C(x, x_n) \end{bmatrix} H^{-1}\bar{t}, \quad \{H\}_{ij} \doteq C(x_i, x_j) + \sigma^2 \delta_{ij}$$

where $H$ is a $n \times n$ matrix and $\delta_{ij}$ is the Kronecker delta.

In this work we are interested in approximating $\hat{f}$ in a suitable $m$-dimensional space that we are going to define. Consider the Mercer-Hilbert expansion of $C(\xi_1, \xi_2)$

$$\int_{\mathbb{R}^d} C(\xi_1, \xi_2)\varphi_i(\xi_2)p(\xi_2)d\xi_2 = \lambda_i\varphi_i(\xi_1), \quad \int_{\mathbb{R}^d} \varphi_i(\xi)\varphi_j(\xi)p(\xi)d\xi = \delta_{ij} \qquad (1)$$

$$C(\xi_1, \xi_2) = \sum_{i=1}^{+\infty} \lambda_i\varphi_i(\xi_1)\varphi_i(\xi_2),$$

where the eigenvalues $\lambda_i$ are ordered in a decreasing way.

Then, in (Zhu et al., 1997) is shown that, at least asymptotically, the optimal model belongs to $\mathcal{M} = \mathrm{Span}\{\varphi_i, i = 1, \ldots, m\}$. This motivates the choice of this space even when dealing with a finite amount of data.

Now we introduce the finite-dimensional approximator which we call Projected Bayes Regression.

**Definition 1** *The PBR approximator is $b(x) = k^{'}(x)w$, where $w \doteq \beta A^{-1}\Phi^{'}\bar{t}$, $\beta \doteq 1/\sigma^2$, $A = \left(\Lambda^{-1} + \beta\Phi^{'}\Phi\right)$, $(\Lambda)_{ij} \doteq \lambda_i\delta_{ij}$ and*

$$k(x) \doteq \begin{bmatrix} \varphi_1(x) \\ \vdots \\ \varphi_m(x) \end{bmatrix}, \quad \Phi \doteq \begin{bmatrix} \varphi_1(x_1) & \cdots & \varphi_m(x_1) \\ \vdots & \ddots & \vdots \\ \varphi_1(x_n) & \cdots & \varphi_m(x_n) \end{bmatrix}.$$

The name PBR comes from the fact that $b(x)$ is the GP predictor when using the mis-specified prior

$$\tilde{f}(x) = \sum_{i=1}^{m} w_i\varphi_i(x), \quad w \sim N(0, \Lambda) \tag{2}$$

whose autocovariance function is the projection of $C(\xi_1, \xi_2)$ on $\mathcal{M}$. From the computational point of view, is interesting to note that the calculation of PBR scales with the data as $O(m^2 n)$, assuming that $n \gg m$ (this is the cost of computing the matrix product $A^{-1}\Phi'$).

Throughout the paper the following measures of performance will be extensively used.

**Definition 2** *Let $s(x)$ be a predictor that uses only information from $\mathcal{D}_n$. Then its $\bar{x}$-error and generalization error are respectively defined as*

$$E_s(n, \bar{x}) \doteq E_{t^*, x^*, \bar{t}}\left[(t^* - s(x^*))^2\right], \quad E_s^g(n) \doteq E_{\bar{x}}\left[E_s(n, \bar{x})\right].$$

*An estimator $s^o(x)$ belonging to a class $\mathcal{H}$ is said $\bar{x}$-optimal or simply optimal if, respectively, $E_{s^o}(n, \bar{x}) \leq E_s(n, \bar{x})$ or $E_{s^o}^g(n) \leq E_s^g(n)$, for all the $s(x) \in \mathcal{H}$ and the data sets $\bar{x}$.*

Note that $\bar{x}$-optimality means optimality for each fixed vector $\bar{x}$ of data points. Obviously, if $s^o(x)$ is $\bar{x}$-optimal it is also simply optimal. These definitions are motivated by the fact that for Gaussian process priors over functions and a predictor $s$ that depends linearly on $\bar{t}$, the computation of $E_s(n, \bar{x})$ can be carried out with finite-dimensional matrix calculations (see Lemma 4 below), although obtaining $E_s^g(n)$ is more difficult, as the average over $\bar{x}$ is usually analytically intractable.

## 3 Optimal finite-dimensional models

We start considering two classes of linear approximators, namely $\mathcal{H}_1 \doteq \left\{g(x) = k^{'}(x)L\bar{t}|L \in \mathbb{R}^{m \times n}\right\}$ and $\mathcal{H}_2 \doteq \left\{h(x) = k^{'}(x)F\Phi^{'}\bar{t}|F \in \mathbb{R}^{m \times m}\right\}$, where the matrices $L$ and $F$ are possibly dependent on the $x_i$ samples. We point out that $\mathcal{H}_2 \subset \mathcal{H}_1$ and that the PBR predictor $b(x) \in \mathcal{H}_2$. Our goal is the characterization of the optimal predictors in $\mathcal{H}_1$ and $\mathcal{H}_2$. Before stating the main result, two preliminary lemmas are given. The first one is proved in (Pilz, 1991) while the second follows from a straightforward calculation.

**Lemma 3** *Let $A \in \mathbb{R}^{n \times n}$, $B \in \mathbb{R}^{n \times r}$, $A > 0$. Then it holds that*

$$\inf_{Z \in \mathbb{R}^{r \times n}} \text{Tr}\left[(ZAZ^{'} - ZB - B^{'}Z^{'})\right] = \text{Tr}\left[-B^{'}A^{-1}B\right]$$

*and the minimum is achieved for the matrix $Z^* = B^{'}A^{-1}$.*

**Lemma 4** *Let $g(x) \in \mathcal{H}_1$. Then it holds that*

$$E_g(n, \bar{x}) = \sum_{i=1}^{+\infty} \lambda_i + \sigma^2 + q(L), \quad q(L) \doteq \text{Tr}\left[LHL^{'} - 2L\Phi\Lambda\right].$$

**Proof.** In view of the $\bar{x}$-error definition, setting $r(x^*) = [C(x^*, x_1) \cdots C(x^*, x_n)]'$, it holds

$$
\begin{aligned}
E_{t^*, \bar{t}}\left[(t^* - k'(x^*)L\bar{t})^2\right] &= \sigma^2 + C(x^*, x^*) + k'(x^*)LHL'k(x^*) \\
&\quad -2k'(x^*)Lr(x^*) \\
&= \sigma^2 + C(x^*, x^*) \\
&\quad +\mathrm{Tr}\left[LHL'k(x^*)k'(x^*) - 2Lr(x^*)k'(x^*)\right].
\end{aligned}
\tag{3}
$$

Note that $E_{x^*}\left[k(x^*)k'(x^*)\right] = I_m$, $E_{x^*}\left[r(x^*)k'(x^*)\right] = \Phi\Lambda$, and, from the Mercer-Hilbert expansion (1), $E_{x^*}[C(x^*, x^*)] = \sum_{i=1}^{+\infty}\lambda_i$. Then, taking the mean of (3) w.r.t. $x^*$, the result follows.$\square$

**Theorem 5** *The predictors $g^o(x) \in \mathcal{H}_1$ given by $L = L^o = \Lambda\Phi'H^{-1}$ and $h^o(x) \in \mathcal{H}_2$ given by $F = F^o = \Lambda\Phi'\Phi(\Phi'H\Phi)^{-1}$, $\forall n \geq m$, are $\bar{x}$-optimal. Moreover*

$$
E_{g^o}(n, \bar{x}) = \sum_{i=1}^{+\infty}\lambda_i + \sigma^2 - \mathrm{Tr}\left[\Lambda\Phi'H^{-1}\Phi\Lambda\right]
\tag{4}
$$

$$
E_{h^o}(n, \bar{x}) = \sum_{i=1}^{+\infty}\lambda_i + \sigma^2 - \mathrm{Tr}\left[\Lambda\Phi'\Phi(\Phi'H\Phi)^{-1}\Phi'\Phi\Lambda\right]
$$

**Proof.** We start considering the $g^o(x)$ case. In view of Lemma 4 we need only to minimize $q(L)$ w.r.t. to the matrix $L$. By applying Lemma 3 with $B = \Phi\Lambda$, $A = H > 0$, $Z = L$, one obtains

$$
\arg\min_L q(L) \doteq L^o = \Lambda\Phi'H^{-1} \quad \min_L q(L) = -\mathrm{Tr}\left[\Lambda\Phi'H^{-1}\Phi\Lambda\right]
\tag{5}
$$

so proving the first result. For the second case, we apply Lemma 4 with $L = F\Phi'$ and then perform the minimization of $q(F\Phi')$, w.r.t. the matrix $F$. This can be done as before noting that $\Phi'H^{-1}\Phi > 0$ only when $n \geq m$. $\square$

Note that the only difference between $g^o(x)$ and the GP predictor derives from the approximation of the functions $C(x, x_k)$ with $\sum_{i=1}^{m}\lambda_i\varphi_i(x)\varphi_i(x_k)$. Moreover the complexity of $g^o(x)$ is $O(n^3)$ the same of $\hat{f}(x)$. On the other hand $h^o(x)$ scales as $O(n^2m)$, so having a computational cost intermediate between the GP predictor and PBR. Intuitively, the PBR method is inferior to $h^o$ as it does not take into account the $\bar{x}$ locations in setting up its prior. We can also show that the PBR predictor $b(x)$ and $h^o(x)$ are asymptotically equivalent.

¿From (4) is clear that the explicit evaluations of $E^g_{g^o}(n)$ and $E^g_{h^o}(n)$ are in general very hard problems, because the mean w.r.t. the $x_i$ samples that enters in the $\Phi$ and $H$ matrices. In the remainder of this section we will derive an upper bound on $E^g_{h^o}(n)$. Consider the class of approximators $\mathcal{H}_3 \doteq \left\{u(x) = k'(x)D\Phi'\bar{t}|D \in \mathbb{R}^{m \times m}, (D)_{ij} = d_i\delta_{ij}\right\}$. Because of the inclusions $\mathcal{H}_3 \subset \mathcal{H}_2 \subset \mathcal{H}_1$, if $u^o(x)$ is the $\bar{x}$-optimal predictor in $\mathcal{H}_3$, then $E^g_{g^o}(n) \leq E^g_{h^o}(n) \leq E^g_{u^o}(n)$. Due the diagonal structure of the matrix $D$, an upper bound to $E^g_{u^o}(n)$ may be explicitly computed, as stated in the next Theorem.

**Theorem 6** *The approximator $u^o(x) \in \mathcal{H}_3$ given by*

$$
(D)_{ij} = (D^o)_{ij} = \frac{\left(\Phi'\Phi\Lambda\right)_{ij}}{\left(\Phi'H\Phi\right)_{ij}}\delta_{ij},
\tag{6}
$$

*is $\bar{x}$-optimal. Moreover an upper-bound on its generalization error is given by*

$$E_{u^o}^g \leq \sum_{i=1}^{+\infty} \lambda_i + \sigma^2 - n \sum_{k=1}^{m} q_k \lambda_k, \quad q_k = \frac{\lambda_k}{c_k} \tag{7}$$

$$c_k = (n-1)\lambda_k + \int C(x,x)\varphi_k^2(x)p(x)dx + \sigma^2.$$

**Proof.** In order to find the $\bar{x}$-optimal approximator in $\mathcal{H}_3$, we start applying the Lemma 4 with $L = D\Phi'$. Then we need to minimize

$$q(D\Phi') = \sum_{i=1}^{m} d_i^2 \left(\Phi' H \Phi\right)_{ii} - 2 \sum_{i=1}^{m} d_i \left(\Phi' \Phi \Lambda\right)_{ii}. \tag{8}$$

w.r.t. $d_i$ so obtaining (6). To bound $E_{u^o}^g(n)$, we first compute the generalization error of a generic approximation $u(x)$ that is $E_u^g = \mathbb{E}_{\bar{x}}\left[q(D\Phi')\right] + \sum_{i=1}^{+\infty} \lambda_i + \sigma^2$. After verifying that

$$\mathbb{E}_{\bar{x}}\left[\left(\Phi'\Phi\Lambda\right)_{ii}\right] = \lambda_i n, \quad \mathbb{E}_{\bar{x}}\left[\left(\Phi' H \Phi\right)_{ii}\right] = nc_i,$$

we obtain from (8), assuming the $d_i$ constant,

$$E_u^g = \sum_{i=1}^{+\infty} \lambda_i + \sigma^2 + n \sum_{i=1}^{m} d_i^2 c_i - 2n \sum_{i=1}^{m} d_i \lambda_i.$$

Minimizing $E_u^g$ w.r.t. $d_i$, and recalling that $u^o(x)$ is also simply optimal the formula (7) follows.□ When $C(\xi_1, \xi_2)$ is stationary, the expression of the $c_i$ coefficient becomes simply $c_i = (n-1)\lambda_i + \sum_{i=1}^{+\infty} \lambda_i + \sigma^2$.

**Remark** : A naive approach to estimating the coefficients in the estimator $\sum_{i=1}^{\infty} w_i \phi_i(x)$ would be to set $\hat{w}_i = n^{-1}(\Phi' \underline{t})_i$ as an approximation to the integral $w_i = \int \phi_i(x)f(x)p(x)dx$. The effect of the matrix $D$ is to "shrink" the $\hat{w}_i$'s of the higher-frequency eigenfunctions. If there was no shrinkage it would be necessary to limit $m$ to stop the poorly-determined $w_i$'s from dominating, but equation 7 shows that in fact the upper bound is improved as $m$ increases. (In fact equation 7 can be used as an upper bound on the GP prediction error; it is tightest when $m \to \infty$.) This is consistent with the idea that increasing $m$ under a Bayesian scheme should lead to improved predictions. In practice one would keep $m < n$, otherwise the approximate algorithm would be computationally more expensive than the $O(n^3)$ GP predictor.

## 4 Choosing $m$

For large $n$, we can show that

$$b(x) \simeq \beta \sum_{i=1}^{m} \varphi_i(x) \left(\frac{1}{\lambda_i} + \beta n\right)^{-1} \sum_{j=1}^{n} \varphi_i(x_j)t_j, \tag{9}$$

where $b(x)$ is the PBR approximator of Definition 1. (This arises because the matrix $\Phi'\Phi$ becomes diagonal in the limit $n \to \infty$ due to the orthogonality of the eigenfunctions.)

In equation 9, the factor $(\lambda_i^{-1} + \beta n)^{-1}$ indicates by how much the prior variance of the $i$th eigenfunction $\phi_i$ has been reduced by the observation of the $n$ datapoints. (Note that this expression is exactly the same as the posterior variance of the mean

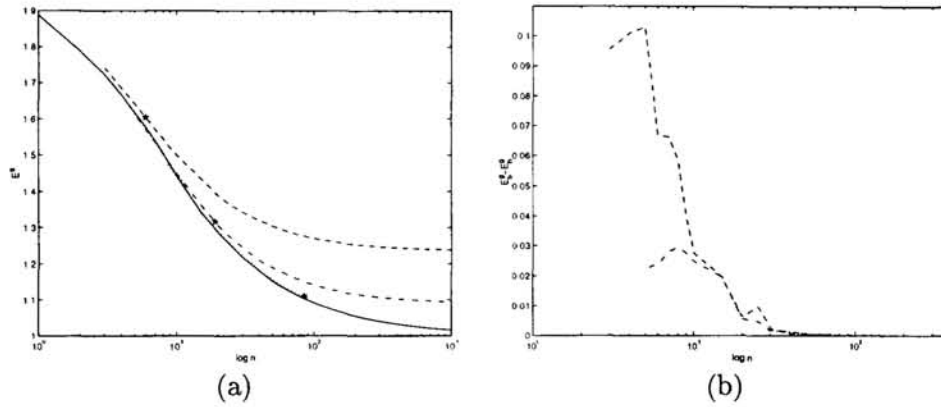

Figure 1: (a) $E^g_{h^o}(n)$ and detaching points for various model orders. Dashed: $m = 3$, dash-dot: $m = 5$, dotted: $m = 8$, solid: $E^g_{\hat{f}}(n)$. (b) $E^g_b(n) - E^g_{h^o}(n)$ plotted against $n$.

of a Gaussian with prior $N(0, \lambda_i)$ given $n$ observations corrupted by Gaussian noise of variance $\beta^{-1}$.) For an eigenfunction with $\lambda_i \gg \sigma^2/n$, the posterior is considerably tighter than the prior, but when $\lambda_i \ll \sigma^2/n$, the prior and posterior have almost the same width, which suggests that there is little point in including these eigenfunctions in the finite-dimensional model. By omitting all but the first $m$ eigenfunctions we add a term $\sum_{i=m+1}^{\infty} \lambda_i$ to the expected generalization error.

This means that for a finite-dimensional model using the first $m$ eigenfunctions, we expect that $E^g_b(n) \simeq E^g_{\hat{f}}(n)$ up to a training set size $\bar{n}$ determined by $\bar{n} = 1/(\beta\lambda_m)$. We call $\bar{n}$ the *detatching point* for the $m$-dimensional approximator. Conversely, in practical regression problems the data set size $n$ is known. Then, from the knowledge of the autocovariance eigenvalues, is possible to determine, via the detatching points formula, the order $m$ of the approximation that should be used in order to guarantee $E^g_{h^o}(n) \simeq E^g_{\hat{f}}(n)$.

## 5 Experimental results

We have conducted experiments using the prior covariance function $C(\xi_1, \xi_2) = (1 + h)e^{-h}$ where $h = |\xi_1 - \xi_2|/\mu$. with $\mu = 0.1$. This covariance function corresponds to a Gaussian process which is once mean-squared differentiable. It lies in the family of stationary covariance functions $C(h) = h^\nu K_\nu(h)$ (where $K_\nu(\cdot)$ is a modified Bessel function), with $\nu = 3/2$. The eigenvalues and eigenfunctions of this covariance kernel for the density $p(x) \sim U(0, 1)$ have been calculated in Vivarelli (1998).

In our first experiment (using $\sigma^2 = 1$) the learning curves of $b(x)$, $h^o(x)$ and $\hat{f}(x)$ were obtained; the average over the choice of training data sets was estimated by using 100 different $\bar{x}$ samples. It was noticed that $E^g_b(n)$ and $E^g_{h^o}(n)$ practically coincide, so only the latter curve is drawn in the pictures.

In Figure 1(a) we have plotted the learning curves for GP regression and the approximation $h^o(x)$ for various model orders. The corresponding detaching points are also plotted, showing their effectiveness in determining the size of data sets for which $E^g_{h^o}(n) \simeq E^g_{\hat{f}}(n)$. The minimum possible error attainable is $\sigma^2 = 1.0$ For finite-dimensional models this is increased by $\sum_{i=m+1}^{\infty} \lambda_i$; these "plateaux" can be clearly seen on the right hand side of Figure 1(a).

Our second experiment demonstrates the differences in performance for the $h^o(x)$ and $b(x)$ estimators, using $\sigma^2 = 0.1$. In Figure 1(b) we have plotted the average difference $E_b^g(n) - E_{h^o}^g(n)$. This was obtained by averaging $E_b(n, \bar{x}) - E_{h^o}(n, \bar{x})$ (computed with the *same* $\bar{x}$, i.e. a paired comparison) over 100 choices of $\bar{x}$, for each $n$. Notice that $h^o$ is superior to the PBR estimator for small $n$ (as expected), but that they are asymptotically equivalent.

## 6   Discussion

In this paper we have shown that a finite-dimensional predictor $h^o$ can be constructed which has lower generalization error than the PBR predictor. Its computational complexity is $O(n^2m)$, lying between the $O(n^3)$ complexity of the GP predictor and $O(m^2n)$ complexity of PBR. We have also shown how to calculate $m$, the number of basis functions required, according to the data set size.

We have used finite-dimensional models to approximate GP regression. An interesting alternative is found in the work of Gibbs and MacKay (1997), where approximate matrix inversion methods that have $O(n^2)$ scaling have been investigated. It would be interesting to compare the relative merits of these two methods.

### Acknowledgements

We thank Francesco Vivarelli for his help in providing the learning curves for $E_f^g(n)$ and the eigenfunctions/values in section 5.

## Footnotes

[1] $O(n^3)$ arises from the inversion of a $n \times n$ matrix.

## References

[1] De Nicolao, G., and Ferrari Trecate, G. (1998). *Identification of NARX models using regularization networks: a consistency result.*.IEEE Int. Joint Conf. on Neural Networks, Anchorage, US, pp. 2407-2412.

[2] Gibbs, M. and MacKay, D. J. C.(1997). *Efficient Implementation of Gaussian Processes.* Cavendish Laboratory, Cambridge, UK. Draft manuscript, available from http://wol.ra.phy.cam.ac.uk/mackay/homepage.html.

[3] Opper, M. (1997). *Regression with Gaussian processes: Average case performance.* In I. K. Kwok-Yee, M. Wong and D.-Y. Yeung (eds), *Theoretical Aspects of Neural Computation: A Multidisciplinary Perspective.* Springer-Verlag.

[4] Pilz, J. (1991). Bayesian estimation and experimental design in linear regression models. Wiley & Sons.

[5] Ripley, B. D. (1996). *Pattern recognition and neural networks.* CUP.

[6] Wahba, G. (1990). *Spline models for observational data.* Society for Industrial and Applied Mathematics. CBMS-NSF Regional Conf. series in applied mathematics.

[7] Whittle, P. (1963). *Prediction and regulation by linear least-square methods.* English Universities Press.

[8] Williams C. K. I. (1998). Prediction with Gaussian processes: from linear regression to linear prediction and beyond. In Jordan, M.I. editor, *Learning and inference in graphical models.* Kluwer Academic Press.

[9] Vivarelli, F. (1998).*Studies on generalization in Gaussian processes and Bayesian Neural Networks.* Forthcoming PhD thesis, Aston University, Birmingham, UK.

[10] Zhu, H., and Rohwer, R. (1996). *Bayesian regression filters and the issue of priors.* Neural Computing and Applications, 4:130-142.

[11] Zhu, H., Williams, C. K. I. Rohwer, R. and Morciniec, M. (1997). *Gaussian regression and optimal finite dimensional linear models.* Tech. Rep. NCRG/97/011. Aston University, Birmingham, UK.